# Generalization in Reinforcement Learning: Successful Examples Using Sparse Coarse Coding

**Richard S. Sutton**
University of Massachusetts
Amherst, MA 01003 USA
rich@cs.umass.edu

## Abstract

On large problems, reinforcement learning systems must use parameterized function approximators such as neural networks in order to generalize between similar situations and actions. In these cases there are no strong theoretical results on the accuracy of convergence, and computational results have been mixed. In particular, Boyan and Moore reported at last year's meeting a series of negative results in attempting to apply dynamic programming together with function approximation to simple control problems with continuous state spaces. In this paper, we present positive results for all the control tasks they attempted, and for one that is significantly larger. The most important differences are that we used sparse-coarse-coded function approximators (CMACs) whereas they used mostly global function approximators, and that we learned online whereas they learned offline. Boyan and Moore and others have suggested that the problems they encountered could be solved by using actual outcomes ("rollouts"), as in classical Monte Carlo methods, and as in the TD($\lambda$) algorithm when $\lambda = 1$. However, in our experiments this always resulted in substantially poorer performance. We conclude that reinforcement learning can work robustly in conjunction with function approximators, and that there is little justification at present for avoiding the case of general $\lambda$.

## 1  Reinforcement Learning and Function Approximation

Reinforcement learning is a broad class of optimal control methods based on estimating value functions from experience, simulation, or search (Barto, Bradtke & Singh, 1995; Sutton, 1988; Watkins, 1989). Many of these methods, e.g., dynamic programming and temporal-difference learning, build their estimates in part on the basis of other

estimates. This may be worrisome because, in practice, the estimates never become exact; on large problems, parameterized function approximators such as neural networks must be used. Because the estimates are imperfect, and because they in turn are used as the targets for other estimates, it seems possible that the ultimate result might be very poor estimates, or even divergence. Indeed some such methods have been shown to be unstable in theory (Baird, 1995; Gordon, 1995; Tsitsiklis & Van Roy, 1994) and in practice (Boyan & Moore, 1995). On the other hand, other methods have been proven stable in theory (Sutton, 1988; Dayan, 1992) and very effective in practice (Lin, 1991; Tesauro, 1992; Zhang & Dietterich, 1995; Crites & Barto, 1996). What are the key requirements of a method or task in order to obtain good performance? The experiments in this paper are part of narrowing the answer to this question.

The reinforcement learning methods we use are variations of the *sarsa* algorithm (Rummery & Niranjan, 1994; Singh & Sutton, 1996). This method is the same as the $TD(\lambda)$ algorithm (Sutton, 1988), except applied to state-action pairs instead of states, and where the predictions are used as the basis for selecting actions. The learning agent estimates action-values, $Q^\pi(s,a)$, defined as the expected future reward starting in state $s$, taking action $a$, and thereafter following policy $\pi$. These are estimated for all states and actions, and for the policy currently being followed by the agent. The policy is chosen dependent on the current estimates in such a way that they jointly improve, ideally approaching an optimal policy and the optimal action-values. In our experiments, actions were selected according to what we call the $\epsilon$-*greedy policy*. Most of the time, the action selected when in state $s$ was the action for which the estimate $\hat{Q}(s,a)$ was the largest (with ties broken randomly). However, a small fraction, $\epsilon$, of the time, the action was instead selected randomly uniformly from the action set (which was always discrete and finite). There are two variations of the sarsa algorithm, one using conventional *accumulate* traces and one using *replace* traces (Singh & Sutton, 1996). This and other details of the algorithm we used are given in Figure 1.

To apply the sarsa algorithm to tasks with a continuous state space, we combined it with a sparse, coarse-coded function approximator known as the CMAC (Albus, 1980; Miller, Gordon & Kraft, 1990; Watkins, 1989; Lin & Kim, 1991; Dean et al., 1992; Tham, 1994). A CMAC uses multiple overlapping tilings of the state space to produce a feature representation for a final linear mapping where all the learning takes place. See Figure 2. The overall effect is much like a network with fixed radial basis functions, except that it is particularly efficient computationally (in other respects one would expect RBF networks and similar methods (see Sutton & Whitehead, 1993) to work just as well). It is important to note that the tilings need not be simple grids. For example, to avoid the "curse of dimensionality," a common trick is to ignore some dimensions in some tilings, i.e., to use hyperplanar slices instead of boxes. A second major trick is "hashing"—a consistent random collapsing of a large set of tiles into a much smaller set. Through hashing, memory requirements are often reduced by large factors with little loss of performance. This is possible because high resolution is needed in only a small fraction of the state space. Hashing frees us from the curse of dimensionality in the sense that memory requirements need not be exponential in the number of dimensions, but need merely match the real demands of the task.

## 2  Good Convergence on Control Problems

We applied the sarsa and CMAC combination to the three continuous-state control problems studied by Boyan and Moore (1995): *2D gridworld*, *puddle world*, and *mountain car*. Whereas they used a model of the task dynamics and applied dynamic programming backups offline to a fixed set of states, we learned *online*, without a model, and backed up whatever states were encountered during complete trials. Unlike Boyan

---

1. Initially: $w_a(f) := \frac{Q_0}{c}$, $e_a(f) := 0$, $\forall a \in Actions$, $\forall f \in CMAC\text{-}tiles$.

2. Start of Trial: $s := random\text{-}state()$;
   $\quad\quad\quad\quad\quad F := features(s)$;
   $\quad\quad\quad\quad\quad a := \epsilon\text{-}greedy\text{-}policy(F)$.

3. Eligibility Traces: $e_b(f) := \lambda e_b(f)$, $\forall b$, $\forall f$;
   3a. Accumulate algorithm: $e_a(f) := e_a(f) + 1$, $\forall f \in F$.
   3b. Replace algorithm: $\quad e_a(f) := 1$, $e_b(f) := 0$, $\forall f \in F$, $\forall b \neq a$.

4. Environment Step:
   Take action $a$; observe resultant reward, $r$, and next state, $s'$.

5. Choose Next Action:
   $F' := features(s')$, unless $s'$ is the terminal state, then $F' := \emptyset$;
   $a' := \epsilon\text{-}greedy\text{-}policy(F')$.

6. Learn: $w_b(f) := w_b(f) + \frac{\alpha}{c}[r + \sum_{f \in F'} w_{a'} - \sum_{f \in F} w_a]e_b(f)$, $\forall b, \forall f$.

7. Loop: $a := a'$; $s := s'$; $F := F'$; if $s'$ is the terminal state, go to 2; else go to 3.

---

Figure 1: The sarsa algorithm for finite-horizon (trial based) tasks. The function $\epsilon$-*greedy-policy*($F$) returns, with probability $\epsilon$, a random action or, with probability $1 - \epsilon$, computes $\sum_{f \in F} w_a$ for each action $a$ and returns the action for which the sum is largest, resolving any ties randomly. The function *features*($s$) returns the set of CMAC tiles corresponding to the state $s$. The number of tiles returned is the constant c. $Q_0$, $\alpha$, and $\lambda$ are scalar parameters.

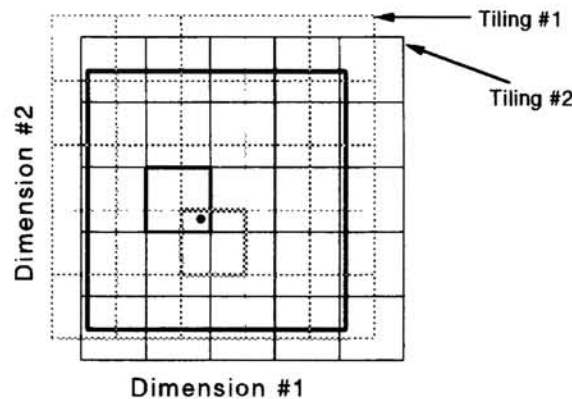

Figure 2: CMACs involve multiple overlapping tilings of the state space. Here we show two $5 \times 5$ regular tilings offset and overlaid over a continuous, two-dimensional state space. Any state, such as that shown by the dot, is in exactly one tile of each tiling. A state's tiles are used to represent it in the sarsa algorithm described above. The tilings need not be regular grids such as shown here. In particular, they are often hyperplanar slices, the number of which grows sub-exponentially with dimensionality of the space. CMACs have been widely used in conjunction with reinforcement learning systems (e.g., Watkins, 1989; Lin & Kim, 1991; Dean, Basye & Shewchuk, 1992; Tham, 1994).

and Moore, we found robust good performance on all tasks. We report here results for the puddle world and the mountain car, the more difficult of the tasks they considered.

Training consisted of a series of trials, each starting from a randomly selected non-goal state and continuing until the goal region was reached. On each step a penalty (negative reward) of $-1$ was incurred. In the puddle-world task, an additional penalty was incurred when the state was within the "puddle" regions. The details are given in the appendix. The 3D plots below show the estimated cost-to-goal of each state, i.e., $\max_a \hat{Q}(s, a)$. In the puddle-world task, the CMACs consisted of 5 tilings, each $5 \times 5$, as in Figure 2. In the mountain-car task we used 10 tilings, each $9 \times 9$.

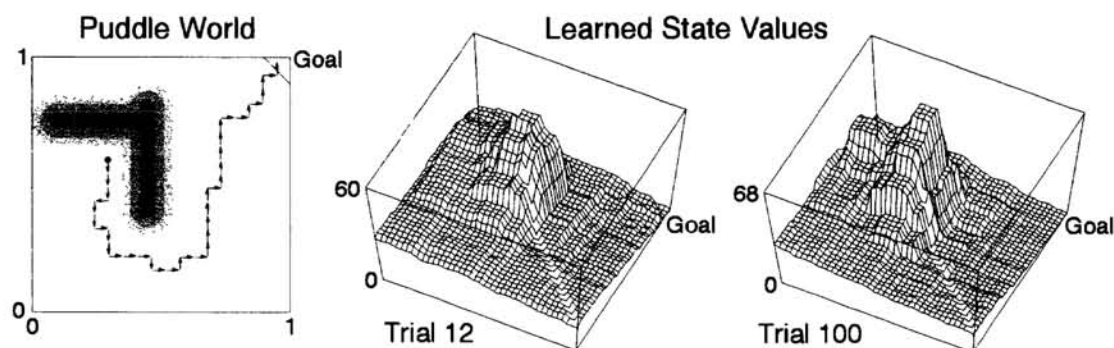

Figure 3: The puddle task and the cost-to-goal function learned during one run.

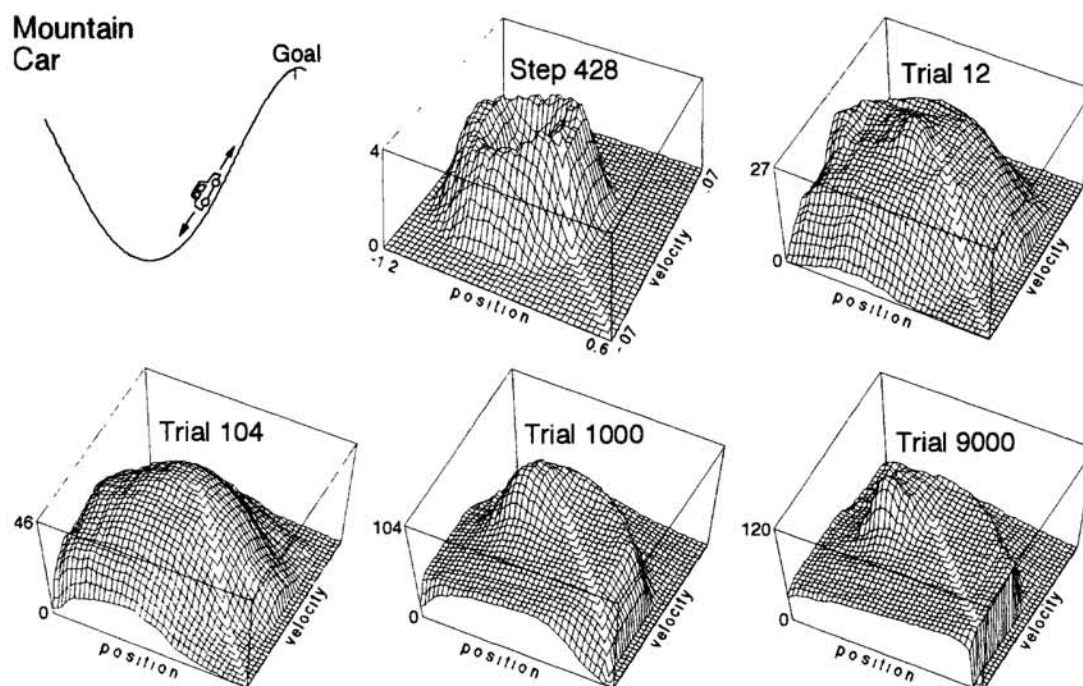

Figure 4: The mountain-car task and the cost-to-goal function learned during one run. The engine is too weak to accelerate directly up the slope; to reach the goal, the car must first move away from it. The first plot shows the value function learned before the goal was reached even once.

We also experimented with a larger and more difficult task not attempted by Boyan and Moore. The *acrobot* is a two-link under-actuated robot (Figure 5) roughly analogous to a gymnast swinging on a highbar (Dejong & Spong, 1994; Spong & Vidyasagar, 1989). The first joint (corresponding to the gymnast's hands on the bar) cannot exert

The object is to swing the endpoint (the feet) above the bar by an amount equal to one of the links. As in the mountain-car task, there are three actions, positive torque, negative torque, and no torque, and reward is −1 on all steps. (See the appendix.)

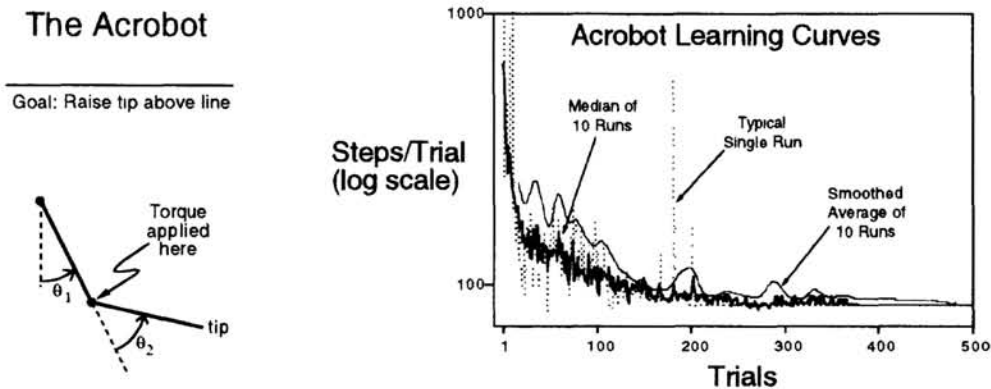

Figure 5: The Acrobot and its learning curves.

## 3  The Effect of $\lambda$

A key question in reinforcement learning is whether it is better to learn on the basis of actual outcomes, as in Monte Carlo methods and as in TD($\lambda$) with $\lambda = 1$, or to learn on the basis of interim estimates, as in TD($\lambda$) with $\lambda < 1$. Theoretically, the former has asymptotic advantages when function approximators are used (Dayan, 1992; Bertsekas, 1995), but empirically the latter is thought to achieve better learning rates (Sutton, 1988). However, hitherto this question has not been put to an empirical test using function approximators. Figures 6 shows the results of such a test.

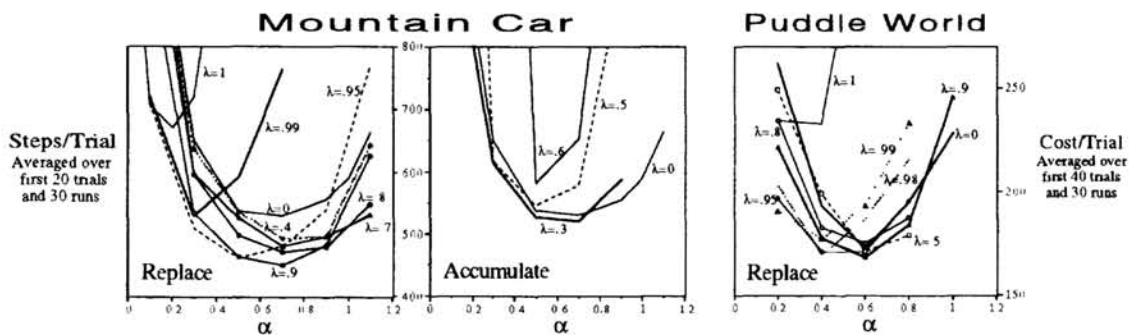

Figure 6: The effects of $\lambda$ and $\alpha$ in the Mountain-Car and Puddle-World tasks.

Figure 7 summarizes this data, and that from two other systematic studies with different tasks, to present an overall picture of the effect of $\lambda$. In all cases performance is an inverted-U shaped function of $\lambda$, and performance degrades rapidly as $\lambda$ approaches 1, where the worst performance is obtained. The fact that performance improves as $\lambda$ is increased from 0 argues for the use of eligibility traces and against 1-step methods such as TD(0) and 1-step Q-learning. The fact that performance improves rapidly as $\lambda$ is reduced below 1 argues against the use of Monte Carlo or "rollout" methods. Despite the theoretical asymptotic advantages of these methods, they are appear to be inferior in practice.

### Acknowledgments
The author gratefully acknowledges the assistance of Justin Boyan, Andrew Moore, Satinder Singh, and Peter Dayan in evaluating these results.

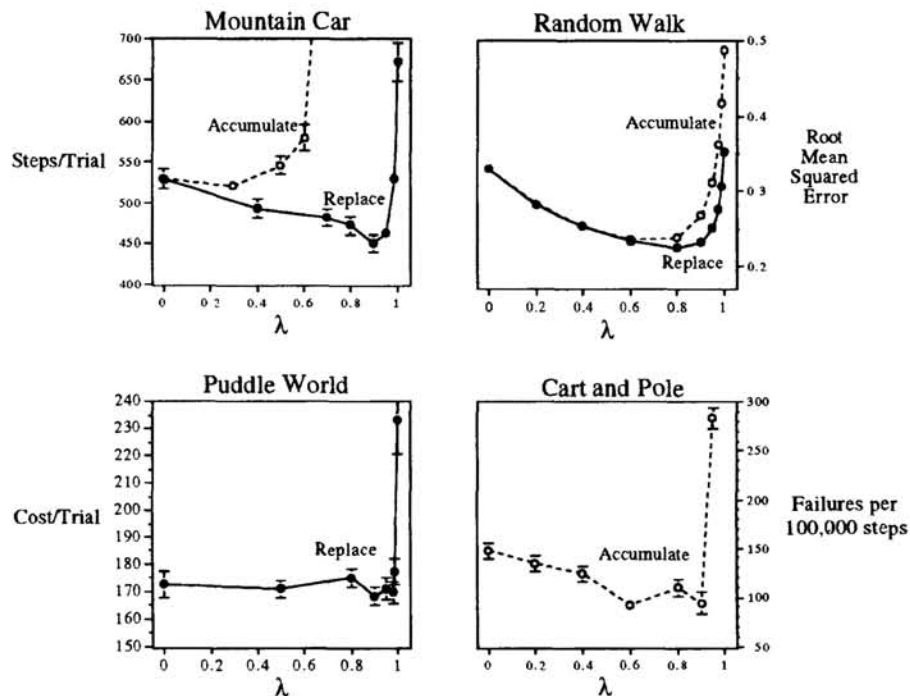

Figure 7: Performance versus $\lambda$, at best $\alpha$, for four different tasks. The left panels summarize data from Figure 6. The upper right panel concerns a 21-state Markov chain, the objective being to predict, for each state, the probability of terminating in one terminal state as opposed to the other (Singh & Sutton, 1996). The lower left panel concerns the pole balancing task studied by Barto, Sutton and Anderson (1983). This is previously unpublished data from an earlier study (Sutton, 1984).

# References

Albus, J. S. (1981) *Brain, Behavior, and Robotics*, chapter 6, pages 139–179. Byte Books.

Baird, L. C. (1995) Residual Algorithms: Reinforcement Learning with Function Approximation. *Proc. ML95*. Morgan Kaufman, San Francisco, CA.

Barto, A. G., Bradtke, S. J., & Singh, S. P. (1995) Real-time learning and control using asynchronous dynamic programming. *Artificial Intelligence*.

Barto, A. G., Sutton, R. S., & Anderson, C. W. (1983) Neuronlike elements that can solve difficult learning control problems. *Trans. IEEE SMC*, *13*, 835–846.

Bertsekas, D. P. (1995) A counterexample to temporal differences learning. *Neural Computation*, *7*, 270–279.

Boyan, J. A. & Moore, A. W. (1995) Generalization in reinforcement learning: Safely approximating the value function. *NIPS-7*. San Mateo, CA: Morgan Kaufmann.

Crites, R. H. & Barto, A. G. (1996) Improving elevator performance using reinforcement learning. *NIPS-8*. Cambridge, MA: MIT Press.

Dayan, P. (1992) The convergence of TD($\lambda$) for general $\lambda$. *Machine Learning*, *8*, 341–362.

Dean, T., Basye, K. & Shewchuk, J. (1992) Reinforcement learning for planning and control. In S. Minton, *Machine Learning Methods for Planning and Scheduling*. Morgan Kaufmann.

DeJong, G. & Spong, M. W. (1994) Swinging up the acrobot: An example of intelligent control. In *Proceedings of the American Control Conference, pages 2158–2162*.

Gordon, G. (1995) Stable function approximation in dynamic programming. *Proc. ML95*.

Lin, L. J. (1992) Self-improving reactive agents based on reinforcement learning, planning and teaching. *Machine Learning*, *8*(3/4), 293–321.

Lin, C-S. & Kim, H. (1991) CMAC-based adaptive critic self-learning control. *IEEE Trans. Neural Networks*, *2*, 530–533.

Miller, W. T., Glanz, F. H., & Kraft, L. G. (1990) CMAC: An associative neural network alternative to backpropagation. *Proc. of the IEEE*, *78*, 1561–1567.

bibliography

Rummery, G. A. & Niranjan, M. (1994) On-line Q-learning using connectionist systems. Technical Report CUED/F-INFENG/TR 166, Cambridge University Engineering Dept.

Singh, S. P. & Sutton, R. S. (1996) Reinforcement learning with replacing eligibility traces. *Machine Learning.*

Spong, M. W. & Vidyasagar, M. (1989) *Robot Dynamics and Control.* New York: Wiley.

Sutton, R. S. (1984) *Temporal Credit Assignment in Reinforcement Learning.* PhD thesis, University of Massachusetts, Amherst, MA.

Sutton, R. S. (1988) Learning to predict by the methods of temporal differences. *Machine Learning, 3*, 9–44.

Sutton, R. S. & Whitehead, S. D. (1993) Online learning with random representations. *Proc. ML93*, pages 314–321. Morgan Kaufmann.

Tham, C. K. (1994) *Modular On-Line Function Approximation for Scaling up Reinforcement Learning.* PhD thesis, Cambridge Univ., Cambridge, England.

Tesauro, G. J. (1992) Practical issues in temporal difference learning. *Machine Learning, 8*(3/4), 257–277.

Tsitsiklis, J. N. & Van Roy, B. (1994) Feature-based methods for large-scale dynamic programming. Techical Report LIDS-P2277, MIT, Cambridge, MA 02139.

Watkins, C. J. C. H. (1989) *Learning from Delayed Rewards.* PhD thesis, Cambridge Univ.

Zhang, W. & Dietterich, T. G., (1995) A reinforcement learning approach to job-shop scheduling. *Proc. IJCAI95.*

## Appendix: Details of the Experiments

In the puddle world, there were four actions, up, down, right, and left, which moved approximately 0.05 in these directions unless the movement would cause the agent to leave the limits of the space. A random gaussian noise with standard deviation 0.01 was also added to the motion along both dimensions. The costs (negative rewards) on this task were $-1$ for each time step plus additional penalties if either or both of the two oval "puddles" were entered. These penalties were -400 times the distance into the puddle (distance to the nearest edge). The puddles were 0.1 in radius and were located at center points (.1, .75) to (.45, .75) and (.45, .4) to (.45, .8). The initial state of each trial was selected randomly uniformly from the non-goal states. For the run in Figure 3, $\alpha = 0.5$, $\lambda = 0.9$, $c = 5$, $\epsilon = 0.1$, and $Q_0 = 0$. For Figure 6, $Q_0 = -20$.

Details of the mountain-car task are given in Singh & Sutton (1996). For the run in Figure 4, $\alpha = 0.5$, $\lambda = 0.9$, $c = 10$, $\epsilon = 0$, and $Q_0 = 0$. For Figure 6, $c = 5$ and $Q_0 = -100$.

In the acrobot task, the CMACs used 48 tilings. Each of the four dimensions were divided into 6 intervals. 12 tilings depended in the usual way on all 4 dimensions. 12 other tilings depended only on 3 dimensions (3 tilings for each of the four sets of 3 dimensions). 12 others depended only on two dimensions (2 tilings for each of the 6 sets of two dimensions. And finally 12 tilings depended each on only one dimension (3 tilings for each dimension). This resulted in a total of $12 \cdot 6^4 + 12 \cdot 6^3 + 12 \cdot 6^2 + 12 \cdot 6 = 18,648$ tiles. The equations of motion were:

$$\ddot{\theta}_1 = -d_1^{-1}(d_2\ddot{\theta}_2 + \phi_1)$$

$$\ddot{\theta}_2 = \left(m_2 l_{c2}^2 + I_2 - \frac{d_2^2}{d_1}\right)^{-1} \left(\tau + \frac{d_2}{d_1}\phi_1 - \phi_2\right)$$

$$d_1 = m_1 l_{c1}^2 + m_2(l_1^2 + l_{c2}^2 + 2l_1 l_{c2}\cos\theta_2) + I_1 + I_2)$$

$$d_2 = m_2(l_{c2}^2 + l_1 l_{c2}\cos\theta_2) + I_2$$

$$\phi_1 = -m_2 l_1 l_{c2}\dot{\theta}_2^2 sin\theta_2 - 2m_2 l_1 l_{c2}\dot{\theta}_2\dot{\theta}_1 sin\theta_2 + (m_1 l_{c1} + m_2 l_1)g\cos(\theta_1 - \pi/2) + \phi_2$$

$$\phi_2 = m_2 l_{c2}g\cos(\theta_1 + \theta_2 - \pi/2)$$

where $\tau \in \{+1, -1, 0\}$ was the torque applied at the second joint, and $\Delta = 0.05$ was the time increment. Actions were chosen after every four of the state updates given by the above equations, corresponding to 5 Hz. The angular velocities were bounded by $\dot{\theta}_1 \in [-4\pi, 4\pi]$ and $\dot{\theta}_2 \in [-9\pi, 9\pi]$. Finally, the remaining constants were $m1 = m2 = 1$ (masses of the links), $l_1 = l_2 = 1$ (lengths of links), $l_{c1} = l_{c2} = 0.5$ (lengths to center of mass of links), $I_1 = I_2 = 1$ (moments of inertia of links), and $g = 9.8$ (gravity). The parameters were $\alpha = 0.2$, $\lambda = 0.9$, $c = 48$, $\epsilon = 0$, $Q_0 = 0$. The starting state on each trial was $\theta_1 = \theta_2 = 0$.